# Bayesian Predictive Profiles with Applications to Retail Transaction Data

**Igor V. Cadez**
Information and Computer Science
University of California
Irvine, CA 92697-3425, U.S.A.
*icadez@ics.uci.edu*

**Padhraic Smyth**
Information and Computer Science
University of California
Irvine, CA 92697-3425, U.S.A.
*smyth@ics.uci.edu*

## Abstract

Massive transaction data sets are recorded in a routine manner in telecommunications, retail commerce, and Web site management. In this paper we address the problem of inferring predictive individual profiles from such historical transaction data. We describe a generative mixture model for count data and use an an approximate Bayesian estimation framework that effectively combines an individual's specific history with more general population patterns. We use a large real-world retail transaction data set to illustrate how these profiles consistently outperform non-mixture and non-Bayesian techniques in predicting customer behavior in out-of-sample data.

## 1   Introduction

Transaction data sets consist of records of pairs of individuals and events, e.g., items purchased (market basket data), telephone calls made (call records), or Web pages visited (from Web logs). Of significant practical interest in many applications is the ability to derive individual-specific (or personalized) models for each individual from the historical transaction data, e.g., for exploratory analysis, adaptive personalization, and forecasting.

In this paper we propose a generative model based on mixture models and Bayesian estimation for learning predictive profiles. The mixture model is used to address the heterogeneity problem: different individuals purchase combinations of products on different visits to the store. The Bayesian estimation framework is used to address the fact that we have different amounts of data for different individuals. For an individual with very few transactions (e.g., only one) we can "shrink" our predictive profile for that individual towards a general population profile. On the other hand, for an individual with many transactions, their predictive model can be more individualized. Our goal is an accurate and computationally efficient modeling framework that smoothly adapts a profile to each individual based on both their own historical data as well as general population patterns. Due to space limitations only selected results are presented here; for a complete description of the methodology and experiments see Cadez et al. (2001).

The idea of using mixture models as a flexible approach for modeling discrete and categorical data has been known for many years, e.g., in the social sciences for latent class analysis (Lazarsfeld and Henry, 1968). Traditionally these methods were only applied to relatively small low-dimensional data sets. More recently there has been a resurgence of interest in mixtures of multinomials and mixtures of conditionally independent Bernoulli models for modeling high-dimensional document-term data in text analysis (e.g., McCallum, 1999; Hoffman, 1999). The work of Heckerman et al. (2000) on probabilistic model-based collaborative filtering is also similar in spirit to the approach described in this paper except that we focus on explicitly extracting individual-level profiles rather than global models (i.e., we have explicit models for each individual in our framework). Our work can be viewed as being an extension of this broad family of probabilistic modeling ideas to the specific case of transaction data, where we deal directly with the problem of making inferences about specific individuals and handling multiple transactions per individual. Other approaches have also been proposed in the data mining literature for clustering and exploratory analysis of transaction data, but typically in a non-probabilistic framework (e.g., Agrawal, Imielinski, and Swami, 1993; Strehl and Ghosh, 2000; Lawrence et al., 2001). The lack of a clear probabilistic semantics (e.g., for association rule techniques) can make it difficult for these models to fully leverage the data for individual-level forecasting.

## 2 Mixture-Basis Models for Profiles

We have an observed data set $D = \{D_1, \ldots, D_N\}$, where $D_i$ is the observed data on the $i$th customer, $1 \leq i \leq N$. Each individual data set $D_i$ consists of one or more transactions for that customer , i.e., $D_i = \{\mathbf{y}_{i1}, \ldots, \mathbf{y}_{ij}, \ldots, \mathbf{y}_{in_i}\}$, where $\mathbf{y}_{ij}$ is the $j$th transaction for customer $i$ and $n_i$ is the total number of transactions observed for customer $i$.

The $j$th transaction for individual $i$, $\mathbf{y}_{ij}$, consists of a description of the set of products (or a "market basket") that was purchased at a specific time by customer $i$ (and $\mathbf{y}_i$ will be used to denote an arbitrary transaction from individual $i$). For the purposes of the experiments described in this paper, each individual transaction $y_{ij}$ is represented as a vector of $d$ counts $\mathbf{y}_{ij} = (n_{ij1}, \ldots n_{ijc}, \ldots, n_{ijC})$, where $n_{ijc}$ indicates how many items of type $c$ are in transaction $\mathbf{y}_{ij}$, $1 \leq c \leq C$.

We define a predictive profile as a probabilistic model $p(\mathbf{y}_i)$, i.e., a probability distribution on the items that individual $i$ will purchase during a store-visit. We propose a simple generative mixture model for an individual's purchasing behavior, namely that a randomly selected transaction $\mathbf{y}_i$ from individual $i$ is generated by one of $K$ components in a $K$-component mixture model. The $k$th mixture component, $1 \leq k \leq K$ is a specific model for generating the counts and we can think of each of the $K$ models as "basis functions" describing prototype transactions. For example, in a clothing store, one might have a mixture component that acts as a prototype for suit-buying behavior, where the expected counts for items such as suits, ties, shirts, etc., given this component, would be relatively higher than for the other items.

There are several modeling choices for the component transaction models for generating item counts. In this paper we choose a particularly simple memoryless multinomial model that operates as follows. Conditioned on $n_{ij}$ (the total number of items in the basket) each of the individual items is selected in a memoryless fashion by $n_{ij}$ draws from a multinomial distribution $P_k = (\theta_{k1}, \ldots, \theta_{kC})$ on the $C$ possible items, $\theta_{kj} = 1$.

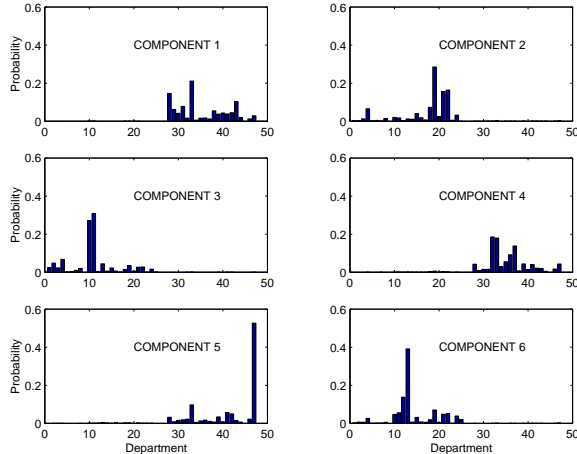

Figure 1: An example of 6 "basis" mixture components fit to retail transaction data.

Figure 1 shows an example of $K = 6$ such basis mixture components that have been learned from a large retail transaction data (more details on learning will be discussed below). Each window shows a different set of component probabilities $P_k$, each modeling a different type of transaction. The components show a striking bimodal pattern in that the multinomial models appear to involve departments that are either above or below department 25, but there is very little probability mass that crosses over. In fact the models are capturing the fact that departments numbered lower than 25 correspond to men's clothing and those above 25 correspond to women's clothing, and that baskets tend to be "tuned" to one set or the other.

## 2.1  Individual-Specific Weights

We further assume that for each individual $i$ there exists a set of $K$ weights, and in the general case these weights are individual-specific, denoted by $\alpha_i = (\alpha_{i1}, \ldots, \alpha_{iK})$, where $\sum_k \alpha_{ik} = 1$. Weight $\alpha_{ik}$ represents the probability that when individual $i$ enters the store their transactions will be generated by component $k$. Or, in other words, the $\alpha_{ik}$'s govern individual $i$'s propensity to engage in "shopping behavior" $k$ (again, there are numerous possible generalizations, such as making the $\alpha_{ik}$'s have dependence over time, that we will not discuss here). The $\alpha_{ik}$'s are in effect the *profile* coefficients for individual $i$, relative to the $K$ component models.

This idea of individual-specific weights (or profiles) is a key component of our proposed approach. The mixture component models $P_k$ are fixed and shared across all individuals, providing a mechanism for borrowing of strength across individual data. The individual weights are in principle allowed to freely vary for each individual within a $K$-dimensional simplex. In effect the $K$ weights can be thought as basis coefficients that represent the location of individual $i$ within the space spanned by the $K$ basis functions (the component $P_k$ multinomials). This approach is quite similar in spirit to the recent probabilistic PCA work of Hofmann (1999) on mixture models for text documents, where he proposes a general mixture model framework that represents documents as existing within a $K$-dimensional simplex of multinomial component models.

The model for each individual is an individual-specific mixture model, where the

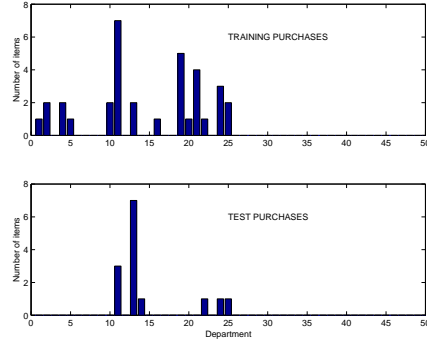

Figure 2: Histograms indicating which products a particular individual purchased, from both the training data and the test data.

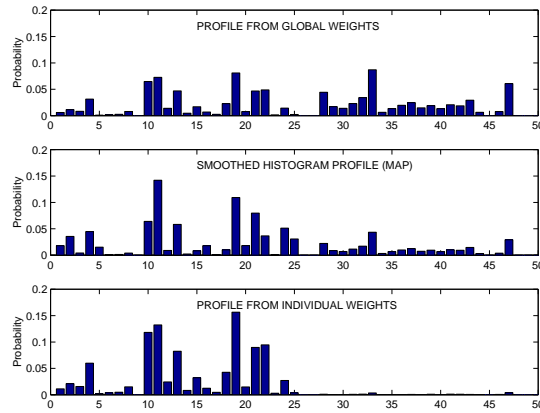

Figure 3: Inferred "effective" profiles from global weights, smoothed histograms, and individual-specific weights for the individual whose data was shown in Figure 2.

weights are specific to individual $i$:

$$
\begin{aligned}
p(\mathbf{y}_{ij}) &= \sum_{k=1}^{K} \alpha_{ik} p(\mathbf{y}_{ij}|k) \\
&= \sum_{k=1}^{K} \alpha_{ik} \prod_{c=1}^{C} \theta_{kc}^{n_{ijc}}.
\end{aligned}
$$

where $\theta_{kc}$ is the probability that the $c$th item is purchased given component $k$ and $n_{ijc}$ is the number of items of category $c$ purchased by individual $i$, during transaction $ij$.

As an example of the application of these ideas, in Figure 2 the training data and test data for a particular individual are displayed. Note that there is some predictability from training to test data, although the test data contains (for example) a purchase in department 14 (which was not seen in the training data). Figure 3 plots the effective profiles[1] for this particular individual as estimated by three different schemes in our modeling approach: (1) global weights that result in everyone

being assigned the same "generic" profile, i.e., $\alpha_{ik} = \alpha_k$, (2) a maximum a posteriori (MAP) technique that smooths each individual's training histogram with a population-based histogram, and (3) individual weights estimated in a Bayesian fashion that are "tuned" to the individual's specific behavior. (More details on each of these methods are provided later in the paper; a complete description can be found in Cadez et al. (2001)).

One can see in Figure 3 that the global weight profile reflects broad population-based purchasing patterns and is not representative of this individual. The smoothed histogram is somewhat better, but the smoothing parameter has "blurred" the individual's focus on departments below 25. The individual-weight profile appears to be a better representation of this individual's behavior and indeed it does provide the best predictive score (of the 3 methods) on the test data in Figure 2. Note that the individual-weights profile in Figure 3 "borrows strength" from the purchases of other similar customers, i.e., it allows for small but non-zero probabilities of the individual making purchases in departments (such as 6 through 9) if he or she has not purchased there in the past. This particular individual's weights, the $\alpha_{ik}$'s, are $(0.00, 0.47, 0.38, 0.00, 0.00.0.15)$ corresponding to the six component models shown in Figure 1. The most weight is placed on components 2, 3 and 6, which agrees with our intuition given the individual's training data.

## 2.2 Learning the Model Parameters

The unknown parameters in our model consist of both the parameters of the $K$ multinomials, $\theta_{kc}, 1 \le k \le K, 1 \le c \le C$, and the vectors of individual-specific profile weights $\boldsymbol{\alpha}_i, 1 \le i \le N$. We investigate two different approaches to learning individual-specific weights:

- **Mixture-Based Maximum Likelihood (ML) Weights:** We treat the weights $\boldsymbol{\alpha}_i$ and component parameters $\theta$ as unknown and use expectation-maximization (EM) to learn both simultaneously. Of course we expect this model to overfit given the number of parameters being estimated but we include it nonetheless as a baseline.

- **Mixture-Based Empirical Bayes (EB) Weights:** We first use EM to learn a mixture of $K$ transaction models (ignoring individuals). We then use a second EM algorithm in weight-space to estimate individual-specific weights $\alpha_i$ for each individual. The second EM phase uses a fixed empirically-determined prior (a Dirichlet) for the weights. In effect, we are learning how best to represent each individual within the $K$-dimensional simplex of basis components. The empirical prior uses the marginal weights ($\boldsymbol{\alpha}$'s) from the first run for the mean of the Dirichlet, and an equivalent sample size of $n = 10$ transactions is used in the results reported in the paper. In effect, this can be viewed as an approximation to either a fully Bayesian hierarchical estimation or an empirical Bayesian approach (see Cadez et al. (2001) for more detailed discussion). We did not pursue the fully Bayesian or empirical Bayesian approaches for computational reasons since the necessary integrals cannot be evaluated in closed form for this model and numerical methods (such as Markov Chain Monte Carlo) would be impractical given the data sizes involved.

We also compare two other approaches for profiling for comparison: (1) **Global Mixture Weights:** instead of individualized weights we set each individual's

---

tion is not a multinomial that can be plotted as a bar chart: however, we can approximate it and we are plotting one such approximation here

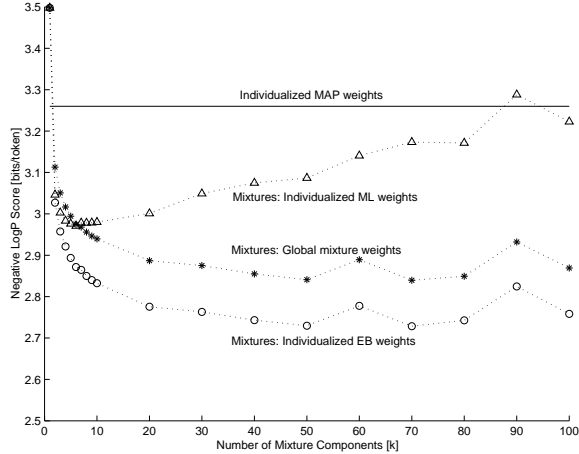

Figure 4: Plot of the negative log probability scores per item (predictive entropy) on out-of-sample transactions, for various weight models as a function of the number of mixture components $K$.

weight vector to the marginal weights ($\alpha_i k = \alpha_k$), and (2) **Individualized MAP weights:** a non-mixture approach where we use an empirically-determined Dirichlet prior directly on the multinomials, and where the equivalent sample size of this prior was "tuned" on the test set to give optimal performance. This provides an (optimistic) baseline of using multinomial profiles directly, without use of any mixture models.

## 3   Experimental Results

To evaluate our approach we used a real-world transaction data set. The data consists of transactions collected at a chain of retail stores over a two-year period. We analyze the transactions here at the store department level (50 categories of items). We separate the data into two time periods (all transactions are time-stamped), with approximately 70% of the data being in the first time period (the training data) and the remainder in the test period data. We train our mixture and weight models on the first period and evaluate our models in terms of their ability to predict transactions that occur in the subsequent out-of-sample test period.

The training data contains data on 4339 individuals, 58,866 transactions, and 164,000 items purchased. The test data consists of 4040 individuals, 25,292 transactions, and 69,103 items purchased. Not all individuals in the test data set appear in the training data set (and vice-versa): individuals in the test data set with no training data are assigned a global population model for scoring purposes.

To evaluate the predictive power of each model, we calculate the log-probability ("logp scores") of the transactions as predicted by each model. Higher logp scores mean that the model assigned higher probability to events that actually occurred. Note that the mean negative logp score over a set of transactions, divided by the total number of items, can be interpreted as a predictive entropy term in bits. The lower this entropy term, the less uncertainty in our predictions (bounded below by zero of course, corresponding to zero uncertainty).

Figure 4 compares the out-of-sample predictive entropy scores as a function of the

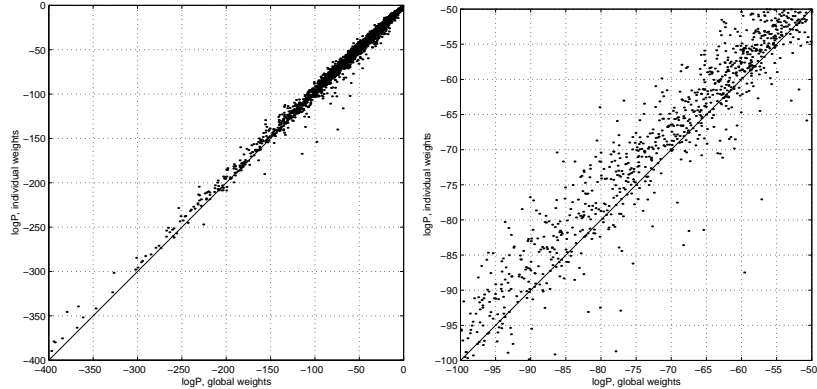

Figure 5: Scatter plots of the log probability scores for each individual on out-of-sample transactions, plotting log probability scores for individual weights versus log probability scores for the global weights model. Left: all data, Right: close up.

number of mixture components $K$ for the mixture-based ML weights, the mixture-based Global weights (where all individuals are assigned the same marginal mixture weights), the mixture-based Empirical Bayes weights, and the non-mixture MAP histogram method (as a baseline). The mixture-based approaches generally outperform the non-mixture MAP histogram approach (solid line). The ML-based mixture weights start to overfit after about 6 mixture components (as expected). The Global mixture weights and individualized mixture weights improve up to about $K = 50$ components and then show some evidence of overfitting. The mixture-based individual weights method is systematically the best predictor, providing a 15% decrease in predictive entropy compared to the MAP histogram method, and a roughly 3% decrease compared to non-individualized global mixture weights.

Figure 5 shows a more detailed comparison of the difference between individual mixtures and the Global profiles, on a subset of individuals. We can see that the Global profiles are systematically worse than the individual weights model (i.e., most points are above the bisecting line). For individuals with the lowest likelihood (lower left of the left plot) the individual weight model is consistently better: typically lower weight *total* likelihood individuals are those with more transactions and items.

In Cadez et al. (2001) we report more detailed results on both this data set and a second retail data set involving 15 million items and 300,000 individuals. On both data sets the individual-level models were found to be consistently more accurate out-of-sample compared to both non-mixture and non-Bayesian approaches. We also found (empirically) that the time taken for EM to converge is roughly linear as both a function of number of components and the number of transactions (plots are omitted due to lack of space), allowing for example fitting of models with 100 mixture components to approximately 2 million baskets in a few hours.

## 4   Conclusions

In this paper we investigated the use of mixture models and approximate Bayesian estimation for automatically inferring individual-level profiles from transaction data records. On a real-world retail data set the proposed framework consistently outperformed alternative approaches in terms of accuracy of predictions on future unseen customer behavior.

**Acknowledgements**

The research described in this paper was supported in part by NSF award IRI-9703120. The work of Igor Cadez was supported by a Microsoft Graduate Research Fellowship.

## Footnotes

[1] We call these "effective profiles" since the predictive model under the mixture assump-

# References

Agrawal, R., Imielenski, T., and Swami, A. (1993) Mining association rules between sets of items in large databases, *Proceedings of the ACM SIGMOD Conference on Management of Data (SIGMOD'98)*, New York: ACM Press, pp. 207–216.

Cadez, I. V., Smyth, P., Ip, E., Mannila, H. (2001) Predictive profiles for transaction data using finite mixture models, Technical Report UCI-ICS-01-67, Information and Computer Science, University of California, Irvine (available online at `www.datalab.uci.edu`.

Heckerman, D., Chickering, D. M., Meek, C., Rounthwaite, R., and Kadie, C. (2000) Dependency networks for inference, collaborative filtering, and data visualization. *Journal of Machine Learning Research*, 1, pp. 49–75.

Hoffmann, T. (1999) Probabilistic latent sematic indexing, *Proceedings of the ACM SIGIR Conference 1999*, New York: ACM Press, 50–57.

Lawrence, R.D., Almasi, G.S., Kotlyar, V., Viveros, M.S., Duri, S.S. (2001) Personalization of supermarket product recommendations, *Data Mining and Knowledge Discovery,* 5 (1/2).

Lazarsfeld, P. F. and Henry, N. W. (1968) *Latent Structure Analysis*, New York: Houghton Mifflin.

McCallum, A. (1999) Multi-label text classification with a mixture model trained by EM, in *AAAI'99 Workshop on Text Learning*.

Strehl, A. and J. Ghosh (2000) Value-based customer grouping from large retail datasets, *Proc. SPIE Conf. on Data Mining and Knowledge Discovery*, SPIE Proc. Vol. 4057, Orlando, pp 33–42.
